# A Multiscale Attentional Framework for Relaxation Neural Networks

**Dimitris I. Tsioutsias**
Dept. of Electrical Engineering
Yale University
New Haven, CT 06520-8285
tsioutsias@cs.yale.edu

**Eric Mjolsness**
Dept. of Computer Science & Engineering
University of California, San Diego
La Jolla, CA 92093-0114
emj@cs.ucsd.edu

## Abstract

We investigate the optimization of neural networks governed by general objective functions. Practical formulations of such objectives are notoriously difficult to solve; a common problem is the poor local extrema that result by any of the applied methods. In this paper, a novel framework is introduced for the solution of large-scale optimization problems. It assumes little about the objective function and can be applied to general nonlinear, non-convex functions; objectives in thousand of variables are thus efficiently minimized by a combination of techniques - deterministic annealing, multiscale optimization, attention mechanisms and trust region optimization methods.

## 1  INTRODUCTION

Many practical problems in computer vision, pattern recognition, robotics and other areas can be described in terms of constrained optimization. In the past decade, researchers have proposed means of solving such problems with the use of neural networks [Hopfield & Tank, 1985; Koch *et al.*, 1986], which are thus derived as relaxation dynamics for the objective functions codifying the optimization task.

One disturbing aspect of the approach soon became obvious, namely the apparent inability of the methods to scale up to practical problems, the principal reason being the rapid increase in the number of local minima present in the objectives as the dimension of the problem increases. Moreover most objectives, $E(v)$, are highly nonlinear, non-convex functions of $v$, and simple techniques (e.g. steepest descent)

will, in general, locate the first minimum from the starting point.

In this work, we propose a framework for solving large-scale instances of such optimization problems. We discuss several techniques which assist in avoiding spurious minima and whose combined result is an objective function solution that is computationally efficient, while at the same time being globally convergent. In section 2.1 we discuss the use of deterministic annealing as a means of avoiding getting trapped into local minima. Section 2.2 describes multiscale representations of the original objective in reduced spatial domains. In section 2.3 we present a scheme for reducing the computational requirements of the optimization method used, by means of a *focus of attention* mechanism. Then, in section 2.4 we introduce a trust region method for the relaxation phase of the framework, which uses second order information (i.e. curvature) of the objective function. In section 3 we present experimental results on the application of our framework to a 2-D region segmentation objective with discontinuities. Finally, section 4 summarizes our presentation.

## 2   THEORETICAL FRAMEWORK

Our optimization framework takes the form of a list of nested loops indicating the order of conceptual (and computational) phases that occur: from the outer to the inner loop we make use of deterministic annealing, a multiscale representation, an attentional mechanism and a trust region optimization method.

### 2.1   ANNEALING NETS

The usefulness of statistical mechanics for designing optimization procedures has recently been established; prime examples are simulated annealing and its various mean field theory approximations [Hopfield & Tank, 1985; Durbin & Willshaw, 1987]. The success of such methods is primarily due to *entropic* terms included in the objective (i.e. *syntactic* terms), but the price to pay is their highly nonlinear form. Interestingly, those terms can effectively be convexified by the use of a "temperature" parameter, $T$, allowing for a reduction in the number of minima and the ability to *track* the solution through "temperature".

### 2.2   MULTISCALE REPRESENTATION

To solve large-scale problems in thousands of variables, we need to speed up the convergence of the method while still retaining *valid* state-space trajectories. To accomplish this we introduce smaller, approximate versions of the problem at coarser spatial scales [Mjolsness *et al.*, 1991]; the nonlinearity of the original objective is maintained at all scales, as opposed to other approaches where the objectives and their derivatives are either approximated by the use of finite difference methods, or solved for by multigrid techniques where a quadratic objective is still assumed. Consequently, the multiscale representation exploits the effective *smoothness* in the objectives: by alternating relaxation phases between coarser and finer scales, we use the former to *identify* extrema and the latter to *localise* them.

### 2.3   FOCUS OF ATTENTION

To further reduce the computational requirements of large-scale optimization (and indirectly control its temporal behavior), we use a *focus of attention* (*FoA*) mechanism [Mjolsness & Miranker, 1993], reminiscent of the *spotlight* hypothesis argued

to exist in early vision systems [Koch & Ullman, 1985; Olshausen *et al.*, 1993]. The effect of a *FoA* is to support efficient, responsive analysis: it allows resources to be *focused* on selected areas of a computation and can rapidly redirect them as the task requirements evolve.

Specifically, the *FoA* becomes a characteristic function, $\pi(\chi)$, determining which of the $N$ neurons are active and which are clamped during relaxation, by use of a discrete-valued vector, $\chi$, and by the rule: $\pi_i(\chi) = 1$ if neuron $v_i$ is in the *FoA*, and zero otherwise. Moreover, a limited number, $n$, of neurons $v_i$ are active at any given instant: $\sum_i \pi_i(\chi) = n$, with $n \ll N$ and $n$ chosen as an optimal *FoA* size. To tie the attentional mechanism to the multiscale representation, we introduce a partition of the neurons $v_i$ into blocks indexed by $a$ (corresponding to coarse-scale block-neurons), via a sparse rectangular matrix $B_{ia} \in \{0,1\}$ such that $\sum_a B_{ia} = 1$, $\forall i$, with $i = 1, \ldots, N$, $a = 1, \ldots, K$ and $K \ll N$. Then $\pi_i(\chi) = \sum_a B_{ia}\chi_a$, and we use each component of $\chi$ for switching a different block of the partition; thus, a neuron $v_i$ is in the *FoA* iff its coarse scale block $a$ is in the *FoA*, as indicated by $\chi_a$. As a result, our *FoA* need not necessarily have a single region of activity: it may well have a distributed activity pattern as determined by the partitions $B_{ia}$.[1]

*Clocked objective function* notation [Mjolsness & Miranker, 1993] makes the task more apparent: during the active-$\chi$ phase the *FoA* is computed for the next active-$v$ phase, determining the subset of neurons $v_i$ on which optimization is to be carried out. We introduce the quantity $E_{;i}[v] \equiv \frac{\partial E}{\partial v_i}\frac{dv_i}{d\tau_i}$ ($\tau_i$ is a time axis for $v_i$) [Mjolsness & Miranker, 1993] as an estimate of the *predicted* $\Delta E$ arising from each $v_i$ if it joins the *FoA*. For Hopfield/Grossberg dynamics this measure becomes:

$$E_{;i}[v] = -g_i'(g_i^{-1}(v_i))\left(\frac{\partial E}{\partial v_i}\right)^2 \equiv -g_i'(u_i)(E_{,i})^2 \tag{1}$$

with $E_{,i} \stackrel{\text{def}}{=} \nabla_i E$, and $g_i$ the transfer function for neuron $v_i$ (e.g. a sigmoid function). Eq. (1) is used here analogously to *saliency measures* introduced into neurophysiological work [Koch & Ullman, 1985]; we propose it as a global measure of conspicuousness. As a result, attention becomes a k-winner-take-all (*kWTA*) network:

$$E_{block}^{(l)} = [\sum_a \chi_a^{(l)} \sum_i B_{ia}^{(l)} E_{;i}(\bar{v}^{(l)}) + kWTA(\chi^{(l)})] \oplus E(v^{(l)}\{\sum_a B_{ia}^{(l)}\chi_a^{(l)}\}), \tag{2}$$

where $l$ refers to the scale for which the *FoA* is being determined ($l = 1, \ldots, L$), $\oplus$ conforms with the clocked objective notation, and the last summand corresponds to the subspace on which optimization is to be performed, as determined by the current *FoA*.[2] Periodically, an analogous *FoA* through spatial scales is run, allowing re-direction of system resources to the scale which seems to be having the largest combined benefit and cost effect on the optimization [Tsioutsias & Mjolsness, 1995]. The combined effect of multiscale optimization and *FoA* is depicted schematically in Fig. 1: reduced-dimension functionals are created and a *FoA beam* "shines" through scales picking the neurons to work on.

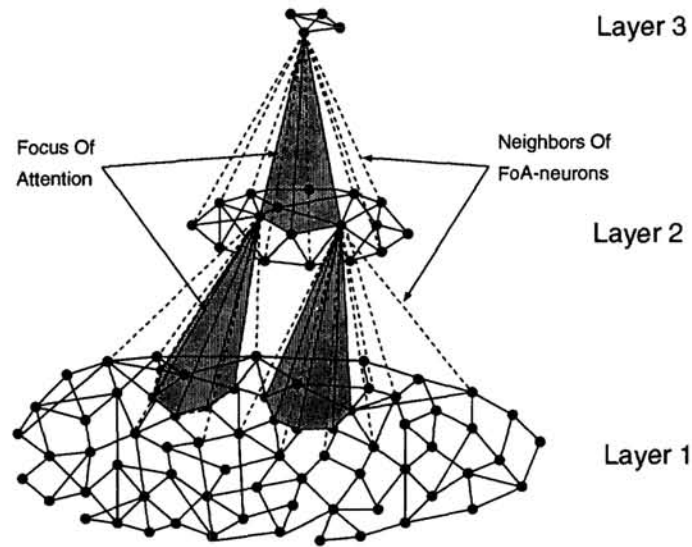

Figure 1: Multiscale Attentional Neural Nets: *FoA* on a layer (e.g. *L*=1) competes with another *FoA* (e.g. *L*=2) to determine both preferable scale and subspace.

## 2.4 OPTIMIZATION PHASE

To overcome the problems generally associated with the steepest descent method, other techniques have been devised. Newton's method, although successful in small to medium-sized problems, does not scale well in large non-convex instances and is computationally intensive. Quasi-Newton methods are efficient to compute, have quadratic termination but are not globally convergent for general nonlinear, non-convex functions. A method that guarantees global convergence is the trust region method [Conn *et al.*, 1993]. The idea is summarized as follows: Newton's method suffers from non-positive definite Hessians; in such a case, the underlying function $m^{(k)}(\delta)$ obtained from the 2nd order Taylor expansion of $E(v_k + \delta)$ does not have a minimum and the method is not defined, or equivalently, the region around the current point $v_k$ in which the Taylor series is adequate does not include a minimizing point of $m^{(k)}(\delta)$. To resolve this, we can define a neighborhood $\Omega_k$ of $v_k$ such that $m^{(k)}(\delta)$ agrees with $E(v_k + \delta)$ in some sense; then, we pick $v_{k+1} = v_k + \delta_k$, where $\delta_k$ minimizes $m^{(k)}(\delta)$, $\forall (v_k + \delta) \in \Omega_k$. Thus, we seek a solution to the resulting subproblem:

$$\min_{\delta} \ m^{(k)}(\delta) \quad s.t. \quad \|\delta\|_p \leq \Delta_k \tag{3}$$

where $\| \cdot \|_p$ is any kind of norm (for instance, the $L_2$ norm leads to the Levenberg-Marquardt methods), and $\Delta_k$ is the radius of $\Omega_k$, adaptively modified based on an *accuracy ratio* $r_k = (\Delta E^{(k)} / \Delta m^{(k)}) \equiv (E^{(k)} - E(v_k + \delta_k))/(m^{(k)}(0) - m^{(k)}(\delta_k))$; $\Delta E^{(k)}$ is the "actual reduction" in $E^{(k)}$ when step $\delta_k$ is taken, and $\Delta m^{(k)}$ the "predicted reduction". The closer $r_k$ is to unity, the better the agreement between the local quadratic model of $E^{(k)}$ and the objective itself is, and $\Delta_k$ is modified adaptively to reflect this [Conn *et al.*, 1993].

We need to make some brief points here (a complete discussion will be given elsewhere [Tsioutsias & Mjolsness, 1995]):

- At each spatial scale of our multiscale representation, we optimize the corresponding objective by applying a trust region method. To obtain sufficient relaxation progress as we move through scales we have to maintain meaningful region sizes, $\Delta_k$; to that end we use a criterion based on the curvature of the functionals along a searching direction.

- The dominant relaxation computation within the algorithm is the solution of eq. (3). We have chosen to solve this subproblem with a preconditioned conjugate gradient method (*PCG*) that uses a truncated Newton step to speed up the computation; steps are accepted when a sufficiently good approximation to the quasi-Newton step is found.[3] In our case, the norm in eq. (3) becomes the *elliptical* norm $\|\delta\|_C = \delta^t C \delta$, where a diagonal preconditioner to the Hessian is used as the scaling matrix $C$.

- If the neuronal connectivity pattern of the original objective is sparse (as happens for most practical combinatorial optimization problems), the pattern of the resulting Hessian can readily be represented by sparse static data structures,[4] as we have done within our framework. Moreover, the partition matrices, $B_{ia}$, introduce a moderate fill-in in the coarser objectives and the sparsity of the corresponding Hessians is again taken into account.

## 3   EXPERIMENTS

We have applied our proposed optimization framework to a spatially structured objective from low-level vision, namely smooth 2-D region segmentation with the inclusion of discontinuity detection processes:

$$
\begin{aligned}
E[\boldsymbol{f}, \boldsymbol{l}^v, \boldsymbol{l}^h] \;=\; & A \sum_{ij} (1 - l_{ij}^v)(f_{i+1,j} - f_{ij})^2 + A \sum_{ij} (1 - l_{ij}^h)(f_{i,j+1} - f_{ij})^2 \\
& + \; B \sum_{ij} (f_{ij} - d_{ij})^2 + C \sum_{ij} (l_{ij}^v + l_{ij}^h) + T \sum_{ij} (\phi(l_{ij}^v) + \phi(l_{ij}^h)) \quad (4)
\end{aligned}
$$

where $\boldsymbol{d}$ is the set of image intensities, $\boldsymbol{f}$ is the real-valued smooth surface to be fit to the data, $\boldsymbol{l}^v$ and $\boldsymbol{l}^h$ are the discrete-valued *line processes* indicating a non-zero value in the intensity gradient, and $\phi(x) = -(2g_0)^{-1}[\ln x + \ln(1-x)]$ is a *barrier function* restricting each variable into (0,1) by infinite barriers at the borders. Eq. (4) is a mixed-nonlinear objective involving both continuous and binary variables; our framework optimizes vectors $\boldsymbol{f}$, $\boldsymbol{l}^h$ and $\boldsymbol{l}^v$ *simultaneously* at any given scale as continuous variables, instead of earlier two-step, alternate continuous/discrete-phase approaches [Terzopoulos, 1986].

We have tested our method on gradually increasing objectives, from a "small" size of $N$=12,288 variables for a 64x64 image, up to a large size of $N$=786,432 variables for a 512x512 image; the results seem to coincide with our theoretical expectations: a significant reduction in computational cost was observed and consistent convergence towards the optimum of the objective was found for various numbers of coarse scales and *FoA* sizes. The dimension of the objective at any scale $l$ was chosen via a *power law*: $N^{(L-l+1)/L}$, where $L$ is the total number of scales and $N$ the size of

the original objective.

The effect of our multiscale optimization with and without a *FoA* is shown in Fig. 2 for the 128x128 and the 512x512 nets, where $E(v^*)$ is the best final configuration with a one-level no-*FoA* net, and *cumulative cost* is an accumulated measure in the number of connection updates at each scale; a consistent scale-up in computational efficiency can be noted when $L > 1$, while the cost measure also reflects the relative total wall-clock times needed for convergence. Fig. 3 shows part of a comparative study we made for saliency measures alternative to eq. (1) (e.g. $g_i'|E_{,i}|$), in order to investigate the validity of eq. (1) as a *predictor* of $\Delta E$: the more prominent "linearity" in the left scatterplot seems to justify our choice of saliency.

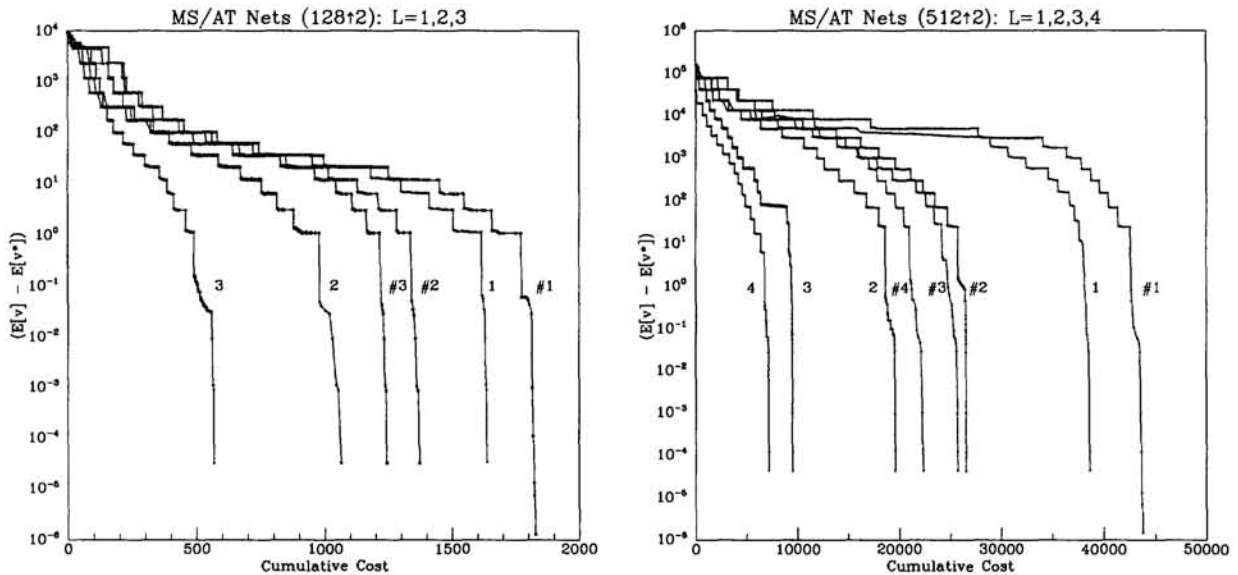

Figure 2: Multiscale Optimization (curves labeled by number of scales used): #-numbered curves correspond to nets without a *FoA*, simply-numbered ones to nets with a *FoA* used at all scales. The lowest costs result from the combined use of multiscale optimization and *FoA*.

## 4   CONCLUSION

We have presented a framework for the optimization of large-scale objective functions using neural networks that incorporate a multiscale attentional mechanism. Our method allows for a continuous adaptation of the system resources to the computational requirements of the relaxation problem through the combined use of several techniques. The framework was applied to a 2-D image segmentation objective with discontinuities; formulations of this problem with tens to hundreds of thousands of variables were then successfully solved.

### Acknowledgements

This work was supported partly by AFOSR-F49620-92-J-0465 and the *Yale Center of Theoretical and Applied Neuroscience.*

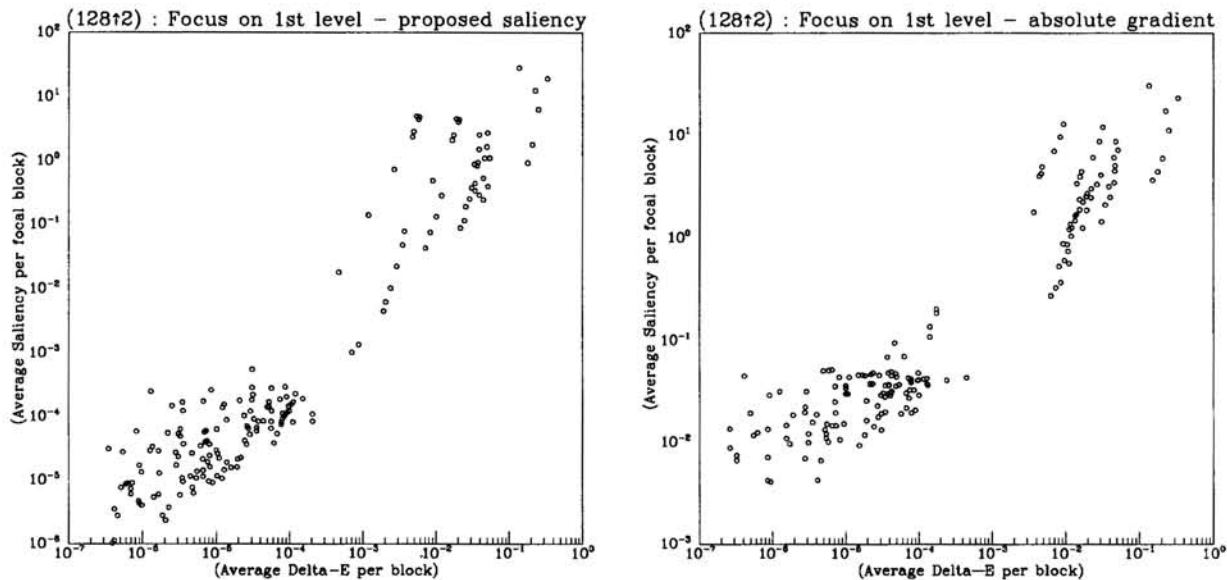

Figure 3: Saliency Comparison: (*left*), saliency as in eq. (1); (*right*), the absolute gradient was used instead.

## Footnotes

[1] Preferably, $B_{ia}$ will be chosen to minimize the number of inter-block connections.

[2] Before computing a new *FoA* we update the neighbors of all neurons that were included in the last focus; this has a similar effect to an implicit *spreading of activation*.

[3] The algorithm can also handle directions of negative curvature.

[4] This property becomes important in a neural net implementation.

## References

A. Conn, N. Gould, A. Sartanaer, & Ph. Toint. (1993) Global Convergence of a Class of Trust Region Algorithms for Optimization Using Inexact Projections on Convex Constraints. *SIAM J. of Optimization*, **3**(1):164-221.

R. Durbin & D. Willshaw. (1987) An Analogue Approach to the TSP Problem Using an Elastic Net Method. *Nature*, **326**:689-691.

J. Hopfield & D. W. Tank. (1985) Neural Computation of Decisions in Optimization Problems. *Biol. Cybernet.*, **52**:141-152.

C. Koch, J. Marroquin & A. Yuille. (1986) Analog 'Neuronal' Networks in Early Vision. *Proc. of the National Academy of Sciences USA*, **83**:4263-4267.

C. Koch, & S. Ullman. (1985) Shifts in Selective Visual Attention: Towards the Underlying Neural Circuitry. *Human Neurobiology*, **4**:219-227.

E. Mjolsness, C. Garrett, & W. Miranker. (1991) Multiscale Optimization in Neural Nets. *IEEE Trans. on Neural Networks*, **2**(2):263-274.

E. Mjolsness & W. Miranker. (1993) Greedy Lagrangians for Neural Networks: Three Levels of Optimization in Relaxation Dynamics. *YALEU/DCS/TR-945*. (URL file://cs.ucsd.edu/pub/emj/papers/yale-TR-945.ps.Z)

B. Olshausen, C. Anderson, & D. Van Essen. (1993) A Neurobiological Model of Visual Attention and Invariant Pattern Recognition Based on Dynamic Routing of Information. *The Journal of Neuroscience*, **13**(11):4700-4719.

D. Terzopoulos. (1986) Regularization of Inverse Visual Problems Involving Discontinuities. *IEEE Trans. PAMI*, **8**:419-429.

D. I. Tsioutsias & E. Mjolsness. (1995) Global Optimization in Neural Nets: A Novel Relaxation Framework. To appear as a *UCSD-CSE-TR*, Dec. 1995.
